# Probabilistic Abstraction Hierarchies

**Eran Segal**
Computer Science Dept.
Stanford University
*eran@cs.stanford.edu*

**Daphne Koller**
Computer Science Dept.
Stanford University
*koller@cs.stanford.edu*

**Dirk Ormoneit**
Computer Science Dept.
Stanford University
*ormoneit@cs.stanford.edu*

## Abstract

Many domains are naturally organized in an abstraction hierarchy or taxonomy, where the instances in "nearby" classes in the taxonomy are similar. In this paper, we provide a general probabilistic framework for clustering data into a set of classes organized as a taxonomy, where each class is associated with a probabilistic model from which the data was generated. The clustering algorithm simultaneously optimizes three things: the assignment of data instances to clusters, the models associated with the clusters, and the structure of the abstraction hierarchy. A unique feature of our approach is that it utilizes global optimization algorithms for both of the last two steps, reducing the sensitivity to noise and the propensity to local maxima that are characteristic of algorithms such as hierarchical agglomerative clustering that only take local steps. We provide a theoretical analysis for our algorithm, showing that it converges to a local maximum of the joint likelihood of model and data. We present experimental results on synthetic data, and on real data in the domains of gene expression and text.

## 1 Introduction

Many domains are naturally associated with a hierarchical taxonomy, in the form of a tree, where instances that are close to each other in the tree are assumed to be more "similar" than instances that are further away. In biological systems, for example, creating a taxonomy of the instances is often one of the first steps in understanding the system. In particular, much of the work on analyzing gene expression data [3] has focused on creating gene hierarchies. Similarly, in text domains, creating a hierarchy of documents is a common task [12, 7].

In many of these applications, the hierarchy is unknown; indeed, discovering the hierarchy is often a key part of the analysis. The standard algorithms applied to the problem typically use an agglomerative bottom-up approach [3] or a divide-and-conquer top-down approach [8]. Although these methods have been shown to be useful in practice, they suffer from several limitations: First, they proceed via a series of local improvements, making them particularly prone to local maxima. Second, at least the bottom-up approaches are typically applied to the raw data; models (if any), are constructed as a post-processing step. Thus, domain knowledge about the type of distribution from which data instances are sampled is rarely used in the formation of the hierarchy.

In this paper, we present *probabilistic abstraction hierarchies (PAH)*, a probabilistically principled general framework for learning abstraction hierarchies from data which overcomes these difficulties. We use a Bayesian approach, where the different models correspond to different abstraction hierarchies. The prior is designed to enforce our intuitions about taxonomies: nearby classes have similar data distributions. More specifically, a model in a PAH is a tree, where each node in the tree is associated with a class-specific probabilistic model (CPM). Data is generated only at the leaves of the tree, so that a model basically defines a mixture distribution whose components are the CPMs at the leaves of

the tree. The CPMs at the internal nodes are used to define the prior over models: We prefer models where the CPM at a child node is close to the CPM at its parent, relative to some distance function between CPMs. Our framework allows a wide range of notions of distance between models; we essentially require only that the distance function be convex in the parameters of the two CPMs. For example, if a CPM is a Gaussian distribution, we might use a simple squared Euclidean distance between the parameters of the two CPMs.

We present a novel algorithm for learning the model parameters and the tree structure in this framework. Our algorithm is based on the structural EM (SEM) approach of [4], but utilizes "global" optimization steps for learning the best possible hierarchy and CPM parameters (see also [5, 13] for similar global optimization steps within SEM). Each step in our procedure is guaranteed to increase the joint probability of model and data, and hence (like SEM) our procedure is guaranteed to converge to a local optimum.

Our approach has several advantages. (1) It provides principled probabilistic semantics for hierarchical models. (2) It is model based, which allows us to exploit domain structural knowledge more easily. (3) It utilizes global optimization steps, which tend to avoid local maxima and help make the model less sensitive to noise. (4) The abstraction hierarchy tends to pull the parameters of one model closer to those of nearby ones, which naturally leads to a form of parameter smoothing or *shrinkage* [12].

We present experiments for PAH on synthetic data and on two real data sets: gene expression and text. Our results show that the PAH approach produces hierarchies that are more robust to noise in the data, and that the learned hierarchies generalize better to test data than those produced by hierarchical agglomerative clustering.

## 2 Probabilistic Abstraction Hierarchy

Let $S$ be the domain of some random observation, e.g., the set of possible assignments to a set of features. Our goal is to take a set of instances in $S$, and to cluster them into some set of $k$ classes. Standard "flat" clustering approaches — for example, Autoclass [1] or the $k$-means algorithm — are special cases of a generative mixture model. In such models, each data instance belongs to one of the $k$ classes, each of which is associated with a different *class-specific probabilistic model* (CPM). Each data instance is sampled independently by first selecting one of the $k$ classes according to a multinomial distribution, and then randomly selecting the data instance itself from the CPM of the chosen class.

In standard clustering models, there is no relation between the individual CPMs, which can be arbitrarily different. In this paper, we propose a model where the different classes are related to each other via an abstraction hierarchy, such that classes that are "nearby" in the hierarchy have similar probabilistic models. More precisely, we define:

**Definition 2.1** *A probabilistic abstraction hierarchy (PAH) $\mathcal{A}$ is a tree $T$ with nodes $V = \{v_1, \ldots, v_m\}$ and undirected edges $E$, such that $T$ has exactly $k$ leaves $v_1, \ldots, v_k$. Each node $v_i$, $i = 1, \ldots, m$, is associated with a CPM $M_i$, which defines a distribution over $S$; we use $\boldsymbol{M}$ to denote $M_1, \ldots, M_m$. We also have a multinomial distribution over the leaves $v_1, \ldots, v_k$; we use $\boldsymbol{\theta}$ to denote the parameters of this distribution.*

Our framework does not, in principle, place restrictions on the form of the CPMs; we can use any probabilistic model that defines a probability distribution over $S$. For example, $M_i$ may be a Bayesian network, in which case its specification would include the parameters, and perhaps also the network structure; in a different setting, $M_i$ may be a hidden Markov model. In practice, however, the choice of CPMs has ramifications both for the overall hierarchical model and the algorithm.

As discussed above, we assume that data is generated only from the leaves of the tree. Thus, we augment $S$ with an additional hidden class variable $C$ for each data item, which takes the values $1, \ldots, k$ denoting the leaf that was chosen to generate this item. Given a PAH $\mathcal{A}$, an element $s \in S$, and a value $c$ for $C$, we define $P(s, c \mid \mathcal{A}) = P(C = c \mid \boldsymbol{\theta})P(s \mid M_c)$, where $P(C = c \mid \boldsymbol{\theta})$ is the multinomial distribution over the leaves and $P(s \mid M_c)$ is the conditional density of the data item given the CPM at leaf $c$. The induced

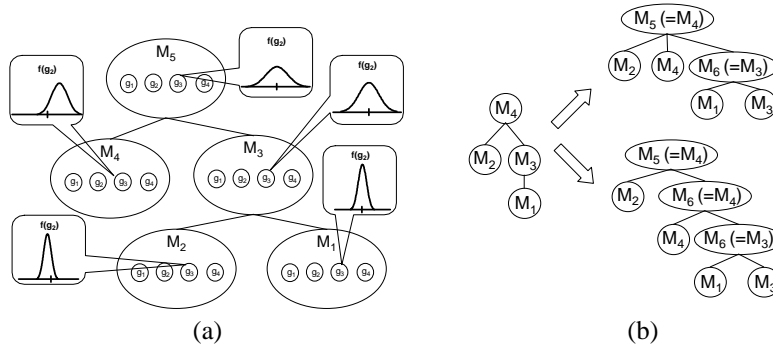

(a)                                                  (b)

Figure 1: (a) A PAH with 3 leaves over a 4-dimensional continuous state space, along with a visualization of the Gaussian distribution for the 3rd dimension. (b) Two different weight-preserving transformations for a tree with 4 leaves $M_1, \ldots, M_4$.

distribution of $s$ given $\mathcal{A}$, from which the data are generated, is simply $P(s \mid \mathcal{A})$, where $c$ is summed out from $P(s, c \mid \mathcal{A})$.

As we mentioned, the role of the internal nodes in the tree is to enforce an intuitive interpretation of the model as an abstraction hierarchy, by enforcing similarity between CPMs at nearby leaves. We achieve this goal by defining a prior distribution over abstraction hierarchies $\mathcal{A}$ that penalizes the distance between neighboring CPMs $M$ and $M'$ using a distance function $\rho(M, M')$. Note that we do not require that $\rho$ be a distance in the mathematical sense; instead, we only require that it be symmetric (as we chose to use undirected trees), non-negative, and that $\rho(M, M') = 0$ iff $M = M'$.[1] One obvious choice is to define $\rho(M, M') = \boldsymbol{D}_{KL}(M; M') + \boldsymbol{D}_{KL}(M'; M)$, where $\boldsymbol{D}_{KL}(M; M')$ is the KL-distance between the distributions that $M$ and $M'$ define over $S$. This distance measure has the advantage of being applicable to any pair of CPMs over the same space, even if their parameterization is different. Given a definition of $\rho$, we define the prior over PAHs as $P(\mathcal{A}) \propto \prod_{(i,j) \in E} \exp\left(-\lambda \rho(M_i, M_j)\right)$, where $\lambda$ represents the extent to which differences in distances are penalized (larger $\lambda$ represents a larger penalty).[2]

Given a set of data instances $D$ with domain $S$, our goal is to find a PAH $\mathcal{A}$ that maximizes $P(\mathcal{A} \mid D) \propto P(\mathcal{A})P(D \mid \mathcal{A})$ or equivalently, $\log P(\mathcal{A}) + \log P(D \mid \mathcal{A})$. By maximizing this expression, we are trading off the fit of the mixture model over the leaves to the data $D$, and the desire to generate a hierarchy in which nearby models are similar. Fig. 1(a) illustrates a typical PAH with Gaussian CPM distributions, where a CPM close to the leaves of the tree is more specialized and thus has fairly peaked distributions. Conversely, CPMs closer to the root of the tree, acting to bridge between their neighbors, are expected to have less peaked distributions and peak only around parts of the distribution which are common to an entire subtree.

## 3   Learning the Models

Our goal in this section is to learn a PAH $\mathcal{A}$ from a data set $D = \{d[1], \ldots, d[N]\}$. This learning task is fairly complex, as many aspects are unknown: the structure of the tree $T$, the CPMs $M_1, \ldots, M_m$ at the nodes of $T$, the parameters $\boldsymbol{\theta}$, and the assignment of the instances in $D$ to leaves of $T$. Hence, the likelihood function has multiple local maxima, and no general method exists for finding the global maximum. In this section, we provide an efficient algorithm for finding a locally optimal $\mathcal{A}$.

To simplify the algorithm, we assume that the structure of the CPMs $M_1, \ldots, M_m$ is fixed. This reduces the choice of each $M_i$ to a pure numerical optimization problem. The general framework of our algorithm extends to cases where we also have to solve the model selection problem for each $M_i$, but the computational issues are somewhat different.

We first discuss the case of complete data, where for each data instance $d[j] \in D$, we are given the leaf from which it was generated. For this case, we show how to learn the structure of the tree $T$ and the setting of the parameters $\theta$ and $M$. This problem, of constructing a tree over a set of points that is not fixed, is very closely related to the *Steiner tree problem* [10], virtually all of whose variants are NP-hard. We propose a heuristic approach that decouples the joint optimization problem into two subproblems: optimizing the CPM parameters given the tree structure, and learning a tree structure given a set of CPMs. Somewhat surprisingly, we show that our careful choice of additive prior allows each of these subproblems to be tackled very effectively using global optimization techniques.

We begin with the task of learning the CPMs. Thus, assume that we are given both the structure of the tree $T$ and the assignment of each data instance $d[m] \in D$ to one of the $k$ leaves, denoted $C[m]$. It remains to find $\theta_{min}, M_{min}$ that minimize $J = -\log P(D \mid \mathcal{A}) - \log P(\mathcal{A})$. Substituting the definitions into $J$, we get that

$$J = -\sum_{m=1}^{|D|} \log P(C[m] \mid \theta) - \sum_i \sum_{m:C[m]=i} \log P(d[m] \mid M_i) + \sum_{(i,j) \in E} \lambda \rho(M_i, M_j). \quad (1)$$

The first term, involving the multinomial parameters $\theta$, separates from the rest, so that the optimization of $J$ relative to $\theta$ reduces to straightforward maximum likelihood estimation. To optimize the CPM parameters, the key property turns out to be the convexity of the $J$ function, which holds in a wide variety of choices of CPMs and $\rho$; in particular, it holds for the models used in our experiments. The convexity property allows us to find the global minimum of $J$ using a simple iterative procedure. In each iteration, we optimize the parameters of one of the $M_i$'s, fixing the parameters of the remaining CPMs $M_j$ ($j \neq i$). This procedure is repeated for each of the $M_i$'s in a round robin fashion, until convergence. By the joint convexity of $J$, this iterative procedure is guaranteed to converge to the global minimum of $J$. An examination of (1) shows that the optimization of each CPM $M_i$ involves only the data cases assigned to $M_i$ (if $i$ is a leaf) and the parameters of the CPMs $M_j$ that are neighbors of $M_i$ in the tree, thereby simplifying the computation substantially.

We now turn our attention to the second subproblem, of learning the structure of the tree given the learned CPMs. We first consider an empty tree containing only the (unconnected) leaf nodes $v_1, \ldots, v_k$, and find the optimal parameter settings for each leaf CPM as described above. Note that these CPMs are unrelated, and the parameters of each one are computed independently of other CPMs. Given this initial set of CPMs for the leaf nodes $v_1, \ldots, v_k$, the algorithm tries to learn a good tree structure $T$ relative to these CPMs. The goal is to find the lowest weight tree, subject to the restriction that the tree structure must keep the same set of leaves $v_1, \ldots, v_k$. Due to the decomposability of $\log P(\mathcal{A})$, the penalty of the tree can be measured via the sum of the edge weights $\rho(M_i, M_j)$. This problem is also a variant of the Steiner tree problem. As a heuristic substitute, we follow the lines of [5] and use a minimum spanning tree (MST) algorithm for constructing low-weight trees.

At each iteration, the algorithm starts out with a tree over some set of nodes $v_1, \ldots, v_m$. It takes the leaves $v_1, \ldots, v_k$ of this tree, and constructs an MST over them. Of course, in the resulting tree, some of the $M_i$ are no longer leaves. This problem is corrected by a transformation that "pushes" a leaf down the tree, duplicating its model; this transformation preserves the weight (score) of the tree. By using only $v_1, \ldots, v_k$, the algorithm simply "throws away" the entire structure of the previous tree. However, we can also construct new MSTs built from all nodes $v_1, \ldots, v_m$ of the previous tree. For all nodes $v_i$ for $1 \leq i \leq k$ which end up as internal nodes, we perform the same transformation described above. In both cases, this transformation is not unique, as it depends on the order in which the steps are executed; see Fig. 1(b). The algorithm therefore generates an entire pool of

candidate trees (from both $v_1, \ldots, v_k$ and $v_1, \ldots, v_m$), generated using different random resolutions of ambiguities in the weight-preserving transformation. For each such tree, the CPM learning algorithm is used to find an optimal setting of the parameters. The trees are evaluated relative to our score ($\log P(\mathcal{A} \mid D)$), and the highest scoring tree is kept.

The tree just constructed has a new set of CPMs, so we can repeat this process. To detect termination, the algorithm also keeps the tree from the previous iteration, and terminates when the score of all trees in the newly constructed pool is lower than the score of the best tree from previous iteration.

Finally, we address the fact that the data we have is incomplete, in that the assignments $C[m]$ of data instances to classes is not determined. We address the problem of incomplete data using the standard *Expectation Maximization (EM)* algorithm [2] and the *structural EM* algorithm [4] which extends EM to the problem of model selection. Starting from an initial model, the algorithm iterates the following two steps: The E-step computes the distribution over the unobserved variables given the observed data and the current model. In our case, the distribution over the unobserved variables is computed by evaluating $P(C[m] = i \mid d[m], \mathcal{A})$ for all $1 \le m \le |D|$. The M-step learns new models that increase the expected log likelihood of the data, relative to the distribution computed in the E-step. In our case, the M-step is precisely the algorithm for complete data described above, but using a soft assignment of data instances to nodes in the tree. The full algorithm is shown in Fig. 2.

A simple analysis along the lines of [4] can be used to show that the log-probability $\log P(\mathcal{A} \mid D)$ increases at every M-step. We therefore obtain the following theorem:

**Theorem 3.1** *The algorithm in Fig. 2 converges to a local maximum of* $\log P(\mathcal{A} \mid D)$.

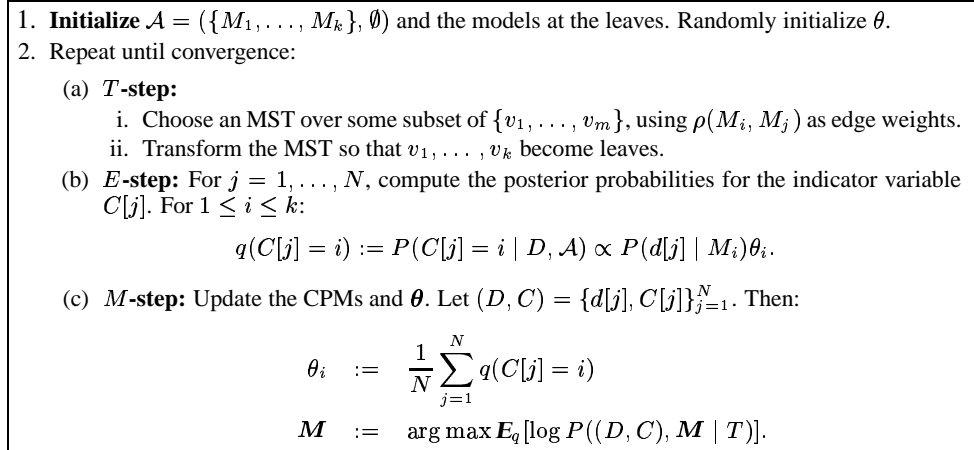

Figure 2: Abstraction Hierarchy Learning Algorithm

## 4   Experimental Results

We focus our experimental results on genomic expression data, although we also provide some results on a text dataset. In gene expression data, the level of mRNA transcript of every gene in the cell is measured simultaneously, using DNA microarray technology. This genomic expression data provides researchers with much insight towards understanding the overall cellular behavior. The most commonly used method for analyzing this data is clustering, a process which identifies clusters of genes that share similar expression patterns (e.g., [3]), and which are therefore also often involved in similar cellular processes. We apply PAH to this data, using CPMs of the form $M_i = \mathcal{N}(\vec{\mu}^i; \sigma^2 I)$, in which case KL-distance is simply: $D_{KL}(M_i; M_j) = \frac{1}{\sigma^2} \sum_{\ell=1}^{n} (\mu_\ell^i - \mu_\ell^j)^2$, which is simply the sum of

squared distances between the means of the corresponding Gaussian components, normalized by their variance. We therefore define $\rho(M_i, M_j) = \boldsymbol{D}_{KL}(M_i; M_j)$.

The most popular clustering method for genomic expression data to date is hierarchical agglomerative clustering (HAC) [3], which builds a hierarchy among the genes by iteratively merging the closest genes relative to some distance metric. We use the same distance metric for HAC. (Note that in HAC the metric is used as the distance between data cases whereas in our algorithm it is used as the distance between models.) To perform a direct comparison between PAH and HAC, we often need to obtain a probabilistic model from HAC. To do so, we create CPMs from the genes that HAC assigned to each internal node. In both PAH and HAC, we then assign each gene (in the training set or the test set) to the hierarchy by choosing the best (highest likelihood) CPM among all the nodes in the tree (including internal nodes) and recording the probability $P(g \mid M_{best})$ that this CPM assigns to the gene.

**Structure Recovery.** A good algorithm for learning abstraction hierarchies should recover the true hierarchy as well as possible. To test this, we generated a synthetic data set, and measured the ability of each method to recover the distances between pairs of instances (genes) in the generating model, where distance here is the length of the path between two genes in the hierarchy.

We generated the data set by sampling from the leaves of a PAH; to make the data realistic, we sampled from a PAH that we learned from a real gene expression data set. To allow a comparison with HAC, we generated one data instance from each leaf. We generated data for 80 (imaginary) genes and 100 experiments, for a total of 8000 measurements. For robustness, we generated 5 different such data sets and ran PAH and HAC for each data set.

We used the correlation and the $L_1$ error between the pairwise distances in the original and the learned tree as measures of similiarity. The correlation was $0.72 \pm 0.08$ for PAH, compared to a much worse $0.27 \pm 0.09$ for HAC. The average $L_1$ error was $4.78 \pm 1.29$ for PAH and $12.46 \pm 1.09$ for HAC. These results show that PAH recovers an abstraction hierarchy much better than HAC.

**Generalization.** We next tested the ability of the different methods to generalize to unobserved (test) data, measuring the extent to which each method captures the underlying structure in the data. We ran these tests on the yeast data set of [6]. We selected 953 genes with significant changes in expression, using their full set of 93 experiments.

Again, we ran PAH and HAC and evaluated performance using 5 fold cross validation. For PAH we also used different settings for $\lambda$ (the coefficient of the penalty term in $P(\mathcal{A})$), which explores the performance in the range of only fitting the data ($\lambda = 0$) and greatly favoring hierarchies in which nearby models are similar (large $\lambda$). In both cases, we learned a model using training data, and evaluated the log-likelihood of test instances as described above. The results, summarized in Fig. 3(a), clearly show that PAH generalizes much better to previously unobserved data than HAC and that PAH works best at some tradeoff between fitting the data and generating a hierarchy in which nearby models are similar.

**Robustness.** Our goal in constructing a hierarchy is to extract meaningful biological conclusions from the data. However, data is invariably partial and noisy. If our analysis produces very different results for slightly different training data, the biological conclusions are unlikely to be meaningful. Thus, we want genes that are assigned to nearby nodes in the tree, to be close together also in hierarchies learned from perturbed data sets.

We tested robustness to noise by learning a model from the original data set and from perturbed data sets in which we permuted a varying percentage of the expression measuments. We then compared the distances (the path length in the tree) between the nodes assigned to every pair of genes in trees learned from the original data and trees learned from perturbed data sets. The results are shown in Fig. 3(b), demonstrating that PAH preserves the pairwise distances extremely well even when $20\%$ of the data is perturbed (and performs reasonably well for $30 - 40\%$ permutation), while HAC completely deteriorates when $20\%$ of the data is permuted.

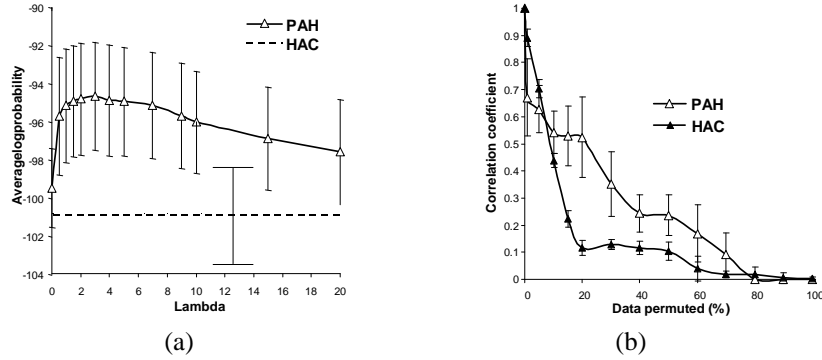

(a)                                        (b)

Figure 3: (a) Generalization to test data (b) Robustness to noise

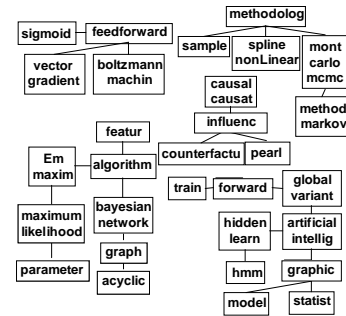

| Model | p | Training set avg. L1 difference | Test set avg. L1 difference |
|-------|-----|------------------|------------------|
| PAH | 90% | 2.57 ± 1.57 | 2.61 ± 1.68 |
| HAC |     | 7.43 ± 4.78 | 11.34 ± 7.2 |
| PAH | 80% | 2.95 ± 1.81 | 2.76 ± 1.61 |
| HAC |     | 9.44 ± 5.68 | 20.87 ± 10.68 |
| PAH | 70% | 3.11 ± 1.84 | 3.17 ± 1.85 |
| HAC |     | 9.79 ± 5.4 | 21.72 ± 13.8 |

(a)                                        (b)

Figure 4: (a) Robustness of PAH and HAC to different subsets of training instances. (b) Word hierarchy learned on Cora data.

A second important test is robustness to our particular choice of training data: a particular training set reflects only a subset of the experiments that we could have performed. In this experiment, we used the Yeast Compendium data of [9], which measures the expression profiles triggered by specific gene mutations. We selected 450 genes and all 298 arrays, focusing on genes that changed significantly. For each of three values of $p$ ranging from 70% to 90%, we generated ten different training sets by sampling (without replacement) $p$ percent of the 450 genes, the rest of which form a test set.

We then placed both training and test genes within the hierarchy. For each data set, every pair of genes either appear together in the training set, the test set, or do not appear together (i.e., one appears in the training set and the other in the test set). We compared, for each pair of genes, their distances in training sets in which they appear together and their distances in test sets in which they appear together. The results are summarized in Fig. 4(a). Our results on the training data show that PAH consistently constructs very similar hierarchies, even from very different subsets of the data. By contrast, the hierarchies constructed by HAC are much less consistent. The results on the test data are even more striking. PAH is very consistent about its classification into the hierachy even of test instances — ones not used to construct the hierarchy. In fact, there is no significant difference between its performance on the training data and the test data. By contrast, HAC places test instances in very different configurations in different trees, reducing our confidence in the biological validity of the learned structure.

**Intuitiveness.** To get qualitative insight into the hierarchies produced, we ran PAH on 350 documents from the Probabilistic Methods category in the Cora dataset (`cora.whizbang.com`) and learned hierarchies among the (stemmed) words. We constructed a vector for each word with an entry for each document whose value is the TFIDF-

weighted frequency of the word within the document. Fig. 4(b) shows parts of the learned hierarchy, consisting of 441 nodes, where we list high confidence words for each node. PAH organized related words into the same region of the tree. Within each region, many words were arranged in a way which is consistent with our intuitive notion of abstraction.

## 5 Discussion

We presented probabilistic abstraction hierarchies, a general framework for learning abstraction hierarchies from data, which relates different classes in the hierarchy by a tree whose nodes correspond to class-specific probability models (CPMs). We utilize a Bayesian approach, where the prior favors hierarchies in which nearby classes have similar data distributions, by penalizing the distance between neighboring CPMs.

A unique feature of PAH is the use of global optimization steps for constructing the hierarchy and for finding the optimal setting of the entire set of parameters. This feature differentiates us from many other approaches that build hierarchies by local improvements of the objective function or approaches that optimize a fixed hierarchy [7]. The global optimization steps help in avoiding local maxima and in reducing sensitivity to noise. Our approach leads naturally to a form of parameter smoothing, and provides much better generalization for test data and robustness to noise than other clustering approaches.

In principle, we can use any probabilistic model for the CPM as long as it defines a probability distribution over the state space. We have recently [14] applied this approach to the substantially more complex problem of clustering proteins based on their amino acid sequence using profile HMMs [11].

**Acknowledgements.** We thank Nir Friedman for useful comments. This work was supported by NSF Grant ACI-0082554 under the NSF ITR program, and by the Sloan Foundation. Eran Segal was also supported by a Stanford Graduate Fellowship (SGF).

## Footnotes

[1]Two models are considered identical if $\forall s \in S : P(s \mid M) = P(s \mid M')$.

[2]Care must be taken to ensure that $P(\mathcal{A})$ is a proper probability distribution, but this will always be the case for the choice of $\rho$ we use in this paper. We also note that, if desired, we can modify this prior to incorporate a prior over the parameters of the $M_i$'s.

## References

[1] P. Cheeseman and J. Stutz. *Bayesian Classification (AutoClass): Theory and Results*. AAAI Press, 1995.

[2] A. P. Dempster, N. M. Laird, and D. B. Rubin. Maximum likelihood from incomplete data via the EM algorithm. *Journal of the Royal Statistical Society*, B 39:1–39, 1977.

[3] M. Eisen, P. Spellman, P. Brown, and D. Botstein. Cluster analysis and display of genome-wide expression patterns. *PNAS*, 95:14863–68, 1998.

[4] N. Friedman. The Bayesian structural EM algorithm. In *Proc. UAI*, 1998.

[5] N. Friedman, M. Ninio, I. Pe'er, and T. Pupko. A structural EM algorithm for phylogentic inference. In *Proc. RECOMB*, 2001.

[6] A.P. Gasch *et al.* Genomic expression program in the response of yeast cells to environmental changes. *Mol. Bio. Cell*, 11:4241–4257, 2000.

[7] T. Hofmann. The cluster-abstraction model: Unsupervised learning of topic hierarchies from text data. In *Proc. IJCAI*, 1999.

[8] T. Hofmann. The cluster-abstraction model: Unsupervised learning of topic hierarchies from text data. In *Proc. International Joint Conference on Artificial Intelligence*, 1999.

[9] T. R. Hughes *et al.* Functional discovery via a compendium of expression profiles. *Cell*, 102(1):109–26, 2000.

[10] F.K. Hwang, D.S.Richards, and P. Winter. *The Steiner Tree Problem*. Annals of Discrete Mathematics, Vol. 53, North-Holland, 1992.

[11] A. Krogh, M. Brown, S. Mian, K. Sjolander, and D. Haussler. Hidden markov models in computational biology: Applications to protein modeling. *Mol. Biology*, 235:1501–1531, 1994.

[12] A. McCallum, R. Rosenfeld, T. Mitchell, and A. Ng. Improving text classification by shrinkage in a hierarchy of classes. In *Proc. ICML*, 1998.

[13] M. Meila and M.I. Jordan. Learning with mixtures of trees. *Machine Learning*, 1:1–48, 2000.

[14] E. Segal and D. Koller. Probabilistic hierarchical clustering for biological data. In *RECOMB*, 2002.
